# Emergence of conjunctive visual features by quadratic independent component analysis

**J.T. Lindgren**
Department of Computer Science
University of Helsinki
Finland
jtlindgr@cs.helsinki.fi

**Aapo Hyvärinen**
HIIT Basic Research Unit
University of Helsinki
Finland
aapo.hyvarinen@cs.helsinki.fi

## Abstract

In previous studies, quadratic modelling of natural images has resulted in cell models that react strongly to edges and bars. Here we apply quadratic Independent Component Analysis to natural image patches, and show that up to a small approximation error, the estimated components are computing conjunctions of two linear features. These conjunctive features appear to represent not only edges and bars, but also inherently two-dimensional stimuli, such as corners. In addition, we show that for many of the components, the underlying linear features have essentially V1 simple cell receptive field characteristics. Our results indicate that the development of the V2 cells preferring angles and corners may be partly explainable by the principle of unsupervised sparse coding of natural images.

## 1 Introduction

Sparse coding of natural images has led to models that resemble the receptive fields in the primate primary visual cortex area V1 (see e.g. [1, 2, 3]). An ongoing research effort is in trying to understand and model the computational principles in visual areas following V1, commonly thought to provide representations for more complicated stimuli. For example, it has recently been shown that in the Macaque monkey, the V2 area following V1 contains neurons responding favourably to angles and corners, but not necessarily to their constituent edges if presented alone [4, 5]. This behaviour can not be easily attained with linear models [6].

In this paper we estimate quadratic models for natural images using Independent Component Analysis (ICA). The used quadratic functions are a natural extension to linear functions (i.e. $\mathbf{l}^T\mathbf{x}$), and give the value of a single feature or component as

$$s = \mathbf{x}^T\mathbf{H}\mathbf{x} + \mathbf{l}^T\mathbf{x}, \tag{1}$$

where the matrix $\mathbf{H}$ specifies weights for second-order interactions between the input variables in stimulus $\mathbf{x}$. This class of functions is equivalent to second-order polynomials of the input, and can compute linear combinations of squared responses of linear models (see e.g. [7]). Another well-known interpretation of components in a quadratic model is as outputs of two-layer neural networks, which is based on an eigenvalue decomposition and will be discussed below.

Estimating a quadratic model for natural images with ICA, we report here the emergence of receptive field models that respond strongly only if the stimulus contains two features that are in a correct spatial arrangement. With a heavy dimensionality reduction, the conjuncted features are mostly collinear (i.e. prefer edges or bars), but with a smaller reduction, additional components emerge that appear to prefer more complex stimuli such as angles or corners. We show that in both cases,

the emerging components approximately operate by computing products between the outputs of two linear submodels that have V1 simple cell characteristics.

The rest of this paper is organized as follows. In section 2 we describe the quadratic ICA in detail. Section 3 outlines the dataset and the preprocessing we used, and section 4 describes the results. Finally, section 5 concludes with discussion and future work.

## 2 Quadratic ICA

Let $\mathbf{x} \in \mathbb{R}^n$ be a vectorized grayscale input image patch. A basic form of linear ICA assumes that each data point is generated as

$$\mathbf{x} = \mathbf{A}\mathbf{s}, \tag{2}$$

where $\mathbf{A}$ is a linear mixing matrix and $\mathbf{s}$ the vector of unknown source signals or independent components. The dimension of $\mathbf{s}$ is assumed to be equal to the dimension of $\mathbf{x}$, possibly after the $\mathbf{x}$ have been reduced by PCA to a smaller dimension. ICA estimation tries to recover $\mathbf{s}$ and the parameter matrix $\mathbf{W} = \mathbf{A}^{-1}$. If the independent components are sparse, this is equivalent to performing sparse coding (for an account of ICA, see e.g. [8]).

It has been proposed that ICA for quadratic models can be performed by first making a quadratic basis expansion on each $\mathbf{x}$ and then applying standard linear ICA [9]. Let the new data vectors $\mathbf{z} \in \mathbb{R}^{n(n+1)/2+n}$ in quadratic space be

$$\mathbf{z} = \Phi([x_1, x_2, ..., x_n]) = [x_1^2, x_1 x_2, ..., x_2^2, x_2 x_3, ..., x_n^2, x_1, x_2, ..., x_n], \tag{3}$$

that is, $\Phi(\mathbf{x})$ generates all the monomials for a second-order polynomial of $\mathbf{x}$, except for the constant term. Such a dimension expansion is also implicit in kernel methods, where a second-order polynomial kernel would be used instead of $\Phi$. Here we work with the more traditional input transformation for simplicity.

From now on, assume that ICA has been performed on the transformed data $\mathbf{z}$. Then the columns $\mathbf{w}_i$ of $\mathbf{W}^T$ make up the quadratic components (cell models, polynomial filters) of the model. As the coefficients in $\mathbf{w}_i$ are weights for a second-order polynomial, it is straightforward to decompose the response $s_i$ of each quadratic component to $\mathbf{x}$ as

$$s_i = \mathbf{w}_i^T \mathbf{z} = \mathbf{x}^T \mathbf{H}_i \mathbf{x} + \mathbf{l}_i^T \mathbf{x}, \tag{4}$$

where $\mathbf{H}_i$ is a symmetric square matrix corresponding to the weights given to all the cross-terms and $\mathbf{l}_i$ weights the first- order monomials. It is well known that the $\mathbf{H}_i$ can be represented in another form by eigenvalue decomposition, leading to the expression

$$s_i = \sum_{j=1}^n \alpha_j (\mathbf{v}_j^T \mathbf{x})^2 + \mathbf{l}_i^T \mathbf{x}, \tag{5}$$

where $\alpha_j$ are the decreasingly sorted eigenvalues of $\mathbf{H}_i$ and $\mathbf{v}_j$ the corresponding eigenvectors. In some cases the representation of eq. 5 can help to understand the model, since the individual eigenvectors can be interpreted as linear receptive fields. A quadratic function in this form is illustrated on the left in figure 1.

However, for our model estimated with quadratic ICA, many of the eigenvectors $\mathbf{v}_j$ did not resemble V1 simple cell receptive fields. Here we propose another decomposition which leads to a simple network computation based on linear receptive fields similar to those of V1 simple cells. Assume that the two eigenvalues of $H_i$ which are largest in absolute value have *opposite signs*; we will refer to these as the dominant eigenvalues and denote them by $\alpha_1$ and $\alpha_n$. This assumption will turn out to hold empirically for our estimated models. Now, including just the two corresponding dominant eigenvectors and ignoring the linear term, we denote

$$\mathbf{v}_+ = (\sqrt{|\alpha_1|}\mathbf{v}_1 + \sqrt{|\alpha_n|}\mathbf{v}_n), \qquad \mathbf{v}_- = (\sqrt{|\alpha_1|}\mathbf{v}_1 - \sqrt{|\alpha_n|}\mathbf{v}_n), \tag{6}$$

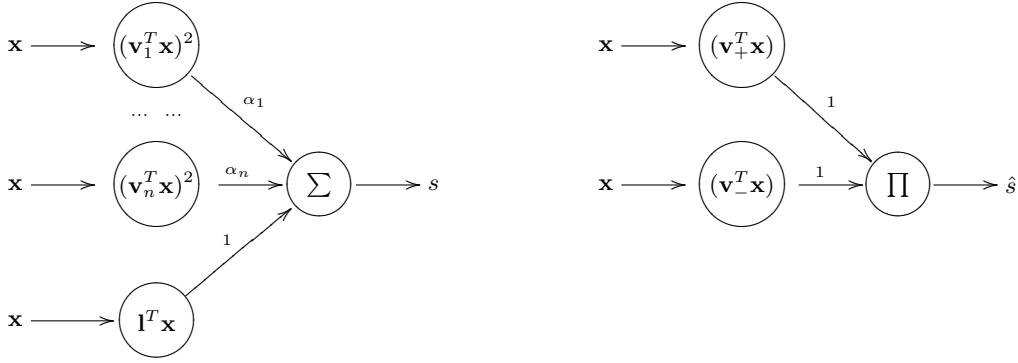

Figure 1: Quadratic components as networks. Using the eigenvalue decomposition, quadratic forms can be interpreted as networks. Left, the computation of a single component $\mathbf{w}$, where the $\mathbf{v}_i$ are the eigenvectors and the $\alpha_i$ the eigenvalues of the matrix $\mathbf{H}$. Right, its product approximation, which is possible if the variance is concentrated on just two eigenvectors with eigenvalues of opposite signs. This turns out to be the case for natural images.

and by using simple arithmetic we obtain a *product approximation* for eq. 5 as

$$\hat{s}_i = \alpha_1(\mathbf{v}_1^T\mathbf{x})^2 + \alpha_n(\mathbf{v}_n^T\mathbf{x})^2 = (\mathbf{v}_+^T\mathbf{x})(\mathbf{v}_-^T\mathbf{x}). \tag{7}$$

This approximation is shown as a network on the right in figure 1, and will be justified later by its relatively small empirical error for our models. Providing that the approximation is good, the intuition is that the component is essentially computing the product of the responses of two linear filters, analogous to a logical AND operation, or a conjunction. We will empirically show that the vectors $\mathbf{v}_+$ and $\mathbf{v}_-$ have resemblance to V1 simple cell receptive fields for our model even if the respective two dominant eigenvectors have more complicated shapes.

## 3   Materials and methods

In our experiments we used the natural image dataset provided by van Hateren and van der Schaaf [2]. This dataset contains over 4000 grayscale images representing natural scenes, each image having a resolution of $1024\times1536$. The intensity distribution over this image set has a very long right tail related to variation in the overall image contrast and intensity [2, 10]. In addition, it is known that the high frequencies in natural images contain sampling artifacts due to rectangular sampling, and that the spectral characteristics are not uniform across frequencies, causing difficulties for gradient-based estimation methods [1]. To alleviate these problems for ICA, we adopt a two-phase preprocessing for the raw images, following [1, 2]. This processing can be considered a very simple model of the physiological pathway containing the retina and the lateral geniculate nucleus (LGN).

First, we address the problem of heavy contrast variation and the long-tailed intensity distribution by taking a natural logarithm of the input images, effectively compressing their dynamic range. This preprocessing is similar to what happens in the first stages of natural visual systems, and has been previously suggested for the current dataset [2]. Next, to correct for the spectral imbalances in the data, we use the whitening filter proposed by Olshausen and Field [1]. This whitening filter cuts the highest frequencies, and balances the frequency distribution otherwise by dampening the dominant low frequencies. We use the filter with the same parameters as in [1]. The whitening filter has bandpass characteristics and hence resembles the center-surround behaviour of LGN cells. In practice, the filtering approximately decorrelates the data.

After preprocessing each image as a whole, we sampled $300,000$ small image patches from the images, each patch having a resolution of $9 \times 9$. Then we subtracted the local DC-component (mean intensity) from each patch. These patches then formed the data we used to estimate the quadratic ICA model. The model fitting was done by transforming the data to the quadratic space using eq. 3, followed by linear ICA. For ICA, we used the FastICA algorithm [11] with $\tanh$ nonlinearity and symmetric estimation of the components. The input dimension in the quadratic space was

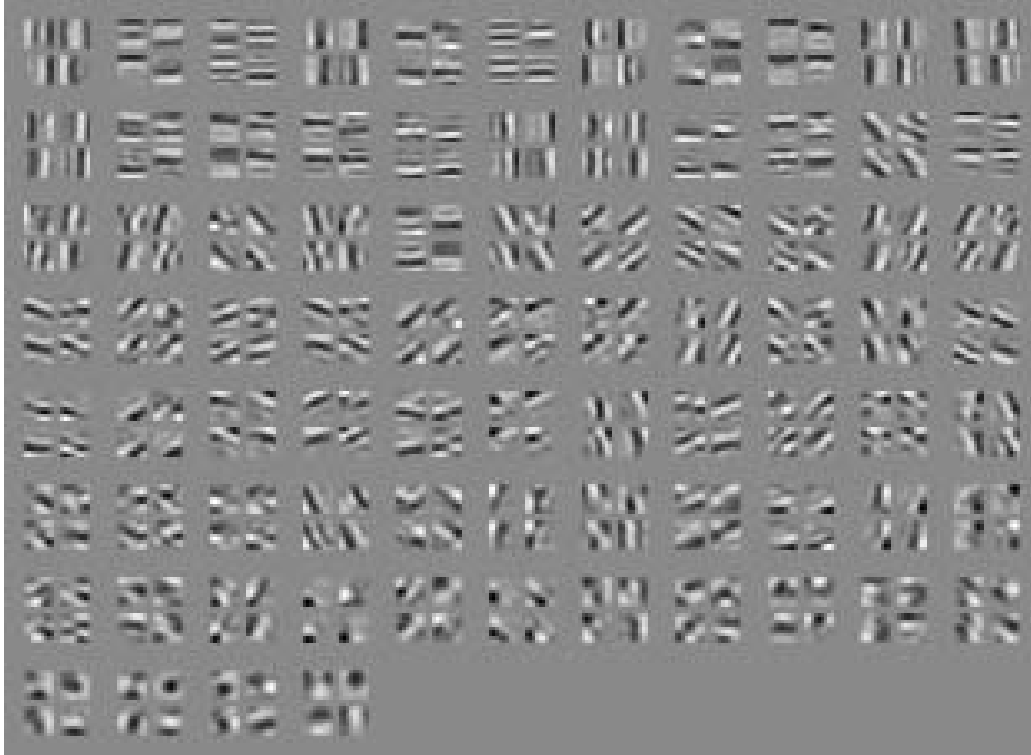

Figure 2: The quadratic ICA components when the model size is very small (81 components). Each quadruple displays the two dominant eigenvectors $\mathbf{v}_1$ and $\mathbf{v}_n$ (top row), and the corresponding vectors $\mathbf{v}_+$ and $\mathbf{v}_-$ (bottom row). Light and dark areas correspond to positive and negative weights, respectively. The components have been sorted by collinearity of the conjuncted features.

$81 * (81 + 1)/2 + 81 = 3402$. We used PCA to drop the dimension by selecting the $400$ most dominant principal axes, covering approximately $50\%$ of the summed eigenvalues. This resulted in estimation of $400$ independent components (or second-order polynomial filters). We also performed additional experiments with $81$ and $1024$ dominant principal axes, corresponding to $18\%$ and $80\%$ coverage. Due to space constraints, we are unable to discuss the $1024$ component model, other than to briefly mention that it conformed to the main results presented in this paper.

To ensure replicable research, the source codes performing the experiments described in this paper have been made publicly available[1].

## 4   Results

In general, interpreting quadratic models can be difficult, and several strategies have been proposed in the literature (see e.g. [12]). However, in the current work the estimated components turned out to be fairly simple (up to a small approximation error, as shown later), and as discussed in section 2, it will be illustrative to display the estimated components in terms of their two dominant eigenvectors $\mathbf{v}_1$ and $\mathbf{v}_n$ of $\mathbf{H}$, and the respective vectors $\mathbf{v}_+$ and $\mathbf{v}_-$ (see eq. 6). Since either pair of the two vectors can be used to compute the approximate component response to any stimuli using eq. 7, the analysis of the components can be based on the vectors $\mathbf{v}_+$ and $\mathbf{v}_-$ if preferred.

Figure 2 shows the quadratic ICA components when a small model was estimated with only 81 components. If we ignore the linear term as in eq. 7, the dominant eigenvectors shown at the top row of each quadruple are equal to the two unit-norm stimuli that the component reacts most highly to (e.g. [12]). Note that the reaction to the eigenvector $\mathbf{v}_n$ (top right) will be highly negative. On the

―――――――――――――
[1]http://www.cs.helsinki.fi/u/jtlindgr/stuff/

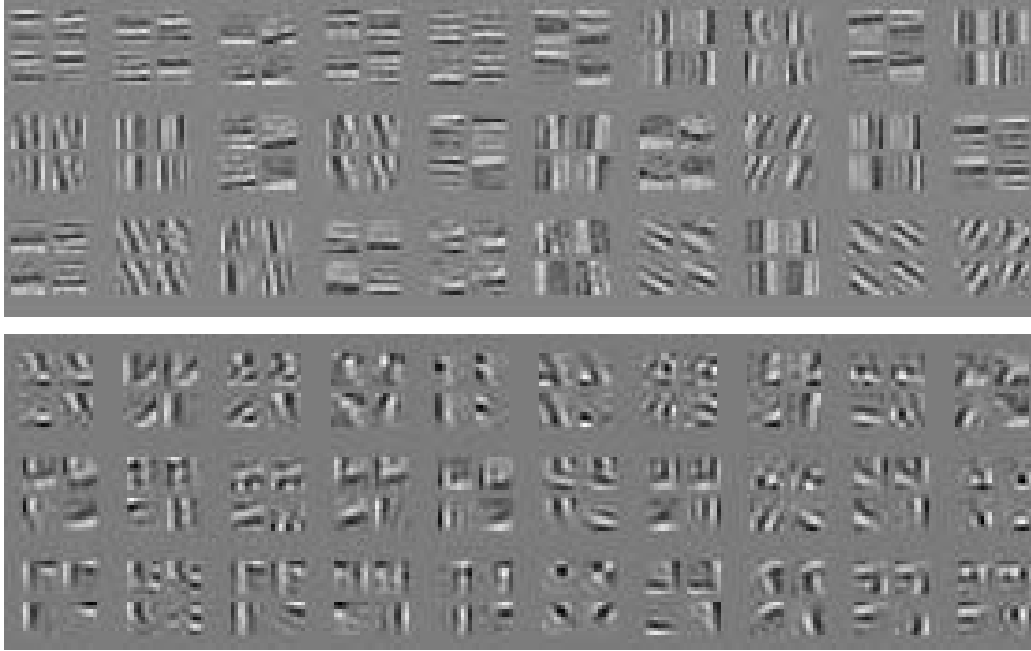

Figure 3: Quadratic ICA components picked from 10 bootstrap iterations with 400 components estimated on each run. All 4000 components were ordered by collinearity of the conjuncted features, and a small sample of each tail is shown. The presentation is the same as in Figure 2. Top, some components that prefer conjunctions of two collinear features. Bottom, components that conjunct two highly orthogonal features. The latter components become more apparent if the model size is large. Clear Gabor-like V1 characteristics can be seen in both cases in the vectors $\mathbf{v}_+$ and $\mathbf{v}_-$, even if the corresponding eigenvectors are more complex.

other hand, both vectors $\mathbf{v}_+$ and $\mathbf{v}_-$ must respond to a stimuli with a non-zero value if the component is to respond strongly. In the case of this small model size, many of the conjuncted features $\mathbf{v}_+$ and $\mathbf{v}_-$ are collinear, and respond strongly to edge- or bar-like stimuli. The feature conjunctions that are not collinear remain more unstructured and appear to react highly to blob-like stimuli. However, both component types are quite different from ordinary linear detectors for edges, bars and blobs, since their conjunctive nature makes them much more selective.

In the following, we will limit the discussion to larger models consisting of 400 components (unless mentioned otherwise). With the higher dimensionality allowed, the diversity of the emerging components increased. Figure 3 shows quadratic ICA components picked from 10 experiments repeated with different subsets of the input patches and different random seeds. Now, in addition to collinear conjunctions (on the top in the image), we also get components that conjunct more orthogonal stimuli (on the bottom). The latter components appear to respond favourably to intuitive visual concepts such as angles and corners. In this case, the benefits of the decomposition to vectors $\mathbf{v}_+$ and $\mathbf{v}_-$ becomes more apparent, as many of the receptive field models retain their resemblance to Gabor-like filters (as in e.g. [1, 2, 8]) even if the corresponding eigenvectors become more complicated.

Next we will validate the above characterization by showing that the approximation of eq. 7 holds up to a generally small error.

First, it turns out that the eigenvalue distributions decay fast for the quadratic forms $\mathbf{H}_i$ of the estimated components. This is illustrated on the left in figure 4, which shows the mean sorted eigenvalues for the 400 components (for a model of 81 components, the figure was similar). Since all the eigenvectors have equal norms, the eigenvalues imply the magnitude of the contribution of its respective eigenvector to the component output value. Due to the fast decay of the eigenvalues, the two dominant eigenvectors are largely responsible for the component output, providing that the linear term $\mathbf{l}$ is insignificant (for some discussion on the linear term in quadratic models, see [12]).

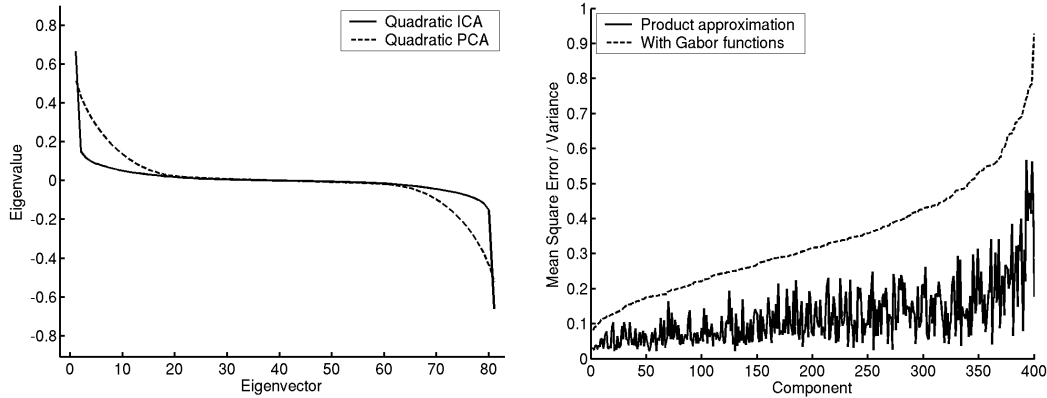

Figure 4: The conjunctive nature of the components is due to the eigenvalues of the quadratic forms $\mathbf{H}_i$ typically emerging as heavily dominated by just two eigenvectors with opposite-sign eigenvalues. This conjunctiveness is further confirmed by the relatively small approximation error caused by ignoring the non-dominant eigenvectors and the linear term. Left, sorted eigenvalues of $\mathbf{H}_i$ averaged over all 400 components for both quadratic ICA and quadratic PCA. It can be seen that the ICA-based eigenvalue distributions tend to decay faster. Right, the relative mean square error of the product approximation for the 400 quadratic ICA components. The components have been sorted by the error of approximation when Gabor functions have been used to model $\mathbf{v}_+$ and $\mathbf{v}_-$.

Here the quadratic part tended to dominate the component responses, which may be because the (co)variances were much larger for the quadratic dimensions of the space than for the linear ones.

The above reasoning is supported by analysis of the prediction error of the product approximation. We examined this by sampling $100,000$ new image patches (not used in the training), and computing the mean square error of the approximation divided by the variance of the component response, i.e. $err(\hat{s}) = E[(s-\hat{s})^2]/Var(s)$. This error is shown on the right in figure 4 for all the 400 components. On average, this relative error was $12\%$ of the respective component variance, ranging from $2\%$ to $57\%$. Hence, the product approximation appears to capture the behaviour of the components rather well. The plot also shows the effect of approximating the vectors $\mathbf{v}_+$ and $\mathbf{v}_-$ with Gabor functions, which are commonly used to model V1 receptive fields. Using Gabor functions, the approximation error increased, ranging from $8\%$ to $93\%$, with mean of $34\%$.

To better understand the obtained error rates, we also fitted linear models to approximate the estimated quadratic cells using least-squares regression. This revealed the quadratic components to have highly nonlinear behaviour. For all components, the error of the linear approximator was over $90\%$, coming close to the baseline $100\%$ error attained if the empirical mean is used as a (constant) predictor.

Since the product approximation only covers the two dominant eigenvectors, it is possible that the rest of the eigenvectors might code for interesting phenomena through further excitatory and inhibitory effects. However, the quick decay of the eigenvalues in our estimated model should make any such effects rather minor. Following the ideas and methods of [12], we explored the possibility that the nondominant eigenvectors coded for invariances of the component. The only strong invariance we found was insensitivity to (possibly local) input sign changes, which is at least partly a structural property of the model, originating from taking squares in eq. 5. In particular, we observed no shift-invariance, consistent with some recent findings in the V2 area of the Macaque monkey [5]. We leave more in-depth exploration of the role of the nondominant eigenvectors as future work.

Finally, we performed some experiments to examine to what extent the method of quadratic ICA on the one hand, and the natural image input data on the other, are responsible for the reported results. For example, it could be argued that the quadratic ICA components might be very similar to the quadratic PCA components. Figure 5 illustrates that this is not trivially so by showing 16 PCA components with large eigenvalues. It can be seen that the PCA components quickly lose resemblance to Gabor-like filters as the eigenvalues decrease. Also, the conjunctive nature of the estimated features is not as clear for the PCA-based components. This is shown on the left in

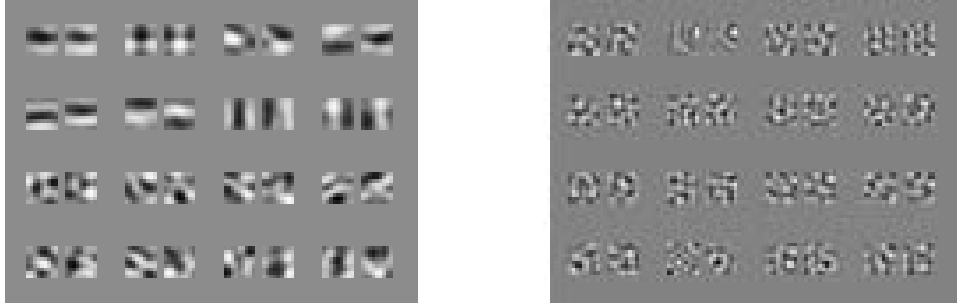

Figure 5: Left, the two top rows show the vectors $\mathbf{v}_+$ and $\mathbf{v}_-$ for the first 8 quadratic PCA components. The two bottom rows display the PCA components $41 - 48$. It can be seen that the PCA components quickly lose any resemblance to Gabor-like receptive fields of V1 simple cells. Right, some typical quadratic ICA components in terms of the vectors $\mathbf{v}_+$ and $\mathbf{v}_-$ when the model was estimated on white noise. The circular shapes are likely artifacts due to the whitening filter.

figure 4, demonstrating that on the average, the eigenvalues of the quadratic forms decay slower for the PCA components. If the whole set of PCA components is studied (not shown), it can be seen that the components appear to change from low-pass filters to high-pass filters as the eigenvalues decrease. Comparing figure 5 to figure 3, both outputs seem characteristic to the method applied, the differences resembling those observed when linear ICA and linear PCA are used to code natural image data.

To verify that the emerging component structures are not artifacts of the modelling methodology, we generated a dataset of artificial images of white noise, having the luminance distribution of the original dataset, but with no spatial or spectral structure. By repeating the model estimation (including preprocessing) on this new dataset, the resulting components did not respond favourably to the same stimuli as before, and they were no longer clearly conjunctive: the eigenvalue distributions decayed fast, but tended to have only one dominant eigenvector. Based on these vectors, the components could be roughly categorized to two classes. The first class responded to spatial forms of center-surround filters, possibly caused by the use of the whitening filter. The second class preferred apparently random configurations of inhibitory and excitatory effects. Some of the components estimated on random data are displayed on the right in figure 5 in terms of vectors $\mathbf{v}_+$ and $\mathbf{v}_-$.

## 5   Discussion

In this paper, we specified a quadratic model for natural images and estimated its parameters with independent component analysis. We reported the emergence of cell models exhibiting strongly non-linear behaviour. In particular, we demonstrated that the estimated cells were essentially computing products between outputs of two linear filters that had V1 simple cell characteristics. Many of these feature conjunctions preferred two collinear features, and yet others corresponded to combinations of more orthogonal stimuli, reacting strongly to angles and corners. Our results indicate that sparse coding of natural images might partially explain the development of angle- or corner-preferring cells in V2.

There has been some previous work describing quadratic models of natural image data (i.e. [13, 7, 9]). Of these, the ICA-based approaches [13, 9] resemble ours the most. Bartsch and Obermayer [13] report curvature detecting cells, but the patch size used and the number of components estimated were very small, making the results inconclusive. Hashimoto [7] sought to replicate complex cell properties with an algorithm based on Kullback-Leibler divergences, and does not report conjunctive features or cells with preferences for angles or corners. Instead, most of the estimated quadratic forms on static image data had only one dominant eigenvector. Our work extends the previous research by reporting the emergence of conjunctive components that combine responses of V1-like linear filters.

The differences of our work to the previous research can be due to various reasons. The number of estimated components (i.e., number of principal components retained) was seen to affect the feature

diversity, and with only $81$ components, the conjuncted features were mostly collinear, producing highly selective edge or bar detectors. Even larger differences to previous work are likely due to different input preprocessing: it is known that unprocessed image data can cause difficulties to statistical estimation of linear models [1, 10] and that both the preprocessing and the size of the used image patches can affect the estimated features [10]. In quadratic modelling, taking products between the dimensions of the input data can cause additional problems for any methods relying on non-robust estimation (such as covariance-based PCA) since the quadratic transform has the strongest boosting effect on outliers and tails of the marginal distributions.

It is worthwhile to note that despite differences to previous work [13, 7, 9], invariances resembling complex cell behaviour did not emerge with our method either, although the class of quadratic models contains the classic energy-detector models of complex cells (fitted in e.g. [3]). It could be that static images and optimization of sparsity alone may not work towards the emergence of invariances, or equivalently, behaviour resembling logical OR operation, unless the model is further constrained (for example as in [3]). Optimizing model likelihood can also be preferable to optimizing output sparseness, but for quadratic ICA it is not clear how to construct a proper probabilistic model.

Finally, an important open question regarding the current work is to what extent the obtained conjunctive features reflect real structures present in the image data. At the time of writing, we have not been able to either prove or disprove the possibility that the pairings are an algorithmic artifact in the following sense: it could be that after the effects of quadratic-space PCA have been accounted for, the quadratic ICA components are only combinations of two rather randomly selected sparse components ($\mathbf{v}_+^T\mathbf{x}$ and $\mathbf{v}_-^T\mathbf{x}$) which are as independent as possible. We are currently investigating this issue.

### Acknowledgments

The authors wish to thank Jarmo Hurri and the anonymous reviewers for helpful comments. This work was supported in part by the IST Programme of the European Community, under the PASCAL Network of Excellence, IST-2002-506778. This publication only reflects the authors' views.

### References

[1] B. A. Olshausen and D. J. Field. Sparse coding with an overcomplete basis set: A strategy employed by V1? *Vision Research*, 37(23):3311–3325, 1997.

[2] J. H. van Hateren and A. van der Schaaf. Independent component filters of natural images compared with simple cells in primary visual cortex. *Proc.R.Soc.Lond. B*, 265:359–366, 1998.

[3] A. Hyvärinen and P. O. Hoyer. Emergence of phase and shift invariant features by decomposition of natural images into independent feature subspaces. *Neural Computation*, 12(7):1705–1720, 2000.

[4] J. Hegdé and D. C. van Essen. Selectivity for complex shapes in primate visual area V2. *The Journal of Neuroscience*, 20(5):RC61–66, 2000.

[5] M. Ito and H. Komatsu. Representation of angles embedded within contour stimuli in area V2 of macaque monkeys. *The Journal of Neuroscience*, 24(13):3313–3324, 2004.

[6] G. Krieger and C. Zetzsche. Nonlinear image operators for the evaluation of local intrisic dimensionality. *IEEE Transactions on Image Processing*, 5(6):1026–1042, 1996.

[7] W. Hashimoto. Quadratic forms in natural images. *Network: Computation in Neural Systems*, 14(4):765–788, 2003.

[8] A. Hyvärinen, J. Karhunen, and E. Oja. *Independent Component Analysis*. Wiley, 2001.

[9] F. Theis and W. Nakamura. Quadratic independent component analysis. *IEICE Trans. Fundamentals*, E87-A(9):2355–2363, 2004.

[10] B. Willmore, P. A. Watters, and D. J. Tolhurst. A comparison of natural-image-based models of simple-cell coding. *Perception*, 29:1017–1040, 2000.

[11] A. Hyvärinen. Fast and robust fixed-point algorithms for independent component analysis. *IEEE Transactions on Neural Networks*, 10(3):626–634, 1999.

[12] P. Berkes and L. Wiskott. On the analysis and interpretation of inhomogeneous quadratic forms as receptive fields. *Neural Computation, accepted*, 2006.

[13] H. Bartsch and K. Obermayer. Second-order statistics of natural images. *Neurocomputing*, 52-54:467–472, 2003.
